# Feature Selection by Maximum Marginal Diversity

**Nuno Vasconcelos**
Department of Electrical and Computer Engineering
University of California, San Diego
nuno@media.mit.edu

## Abstract

We address the question of feature selection in the context of visual recognition. It is shown that, besides efficient from a computational standpoint, the infomax principle is nearly optimal in the minimum Bayes error sense. The concept of marginal diversity is introduced, leading to a generic principle for feature selection (the principle of maximum marginal diversity) of extreme computational simplicity. The relationships between infomax and the maximization of marginal diversity are identified, uncovering the existence of a family of classification procedures for which near optimal (in the Bayes error sense) feature selection does not require combinatorial search. Examination of this family in light of recent studies on the statistics of natural images suggests that visual recognition problems are a subset of it.

## 1 Introduction

It has long been recognized that *feature extraction* and *feature selection* are important problems in statistical learning. Given a classification or regression task in some *observation space* $\mathcal{Z}$ (typically high-dimensional), the goal is to find the best transform $T$ into a *feature space* $\mathcal{X}$ (typically lower dimensional) where learning is easier (e.g. can be performed with less training data). While in the case of feature extraction there are few constraints on $T$, for feature selection the transformation is constrained to be a projection, i.e. the components of a *feature vector* in $\mathcal{X}$ are a subset of the components of the associated vector in $\mathcal{Z}$. Both feature extraction and selection can be formulated as optimization problems where the goal is to find the transform that best satisfies a given criteria for "feature goodness".

In this paper we concentrate on visual recognition, a subset of the classification problem for which various optimality criteria have been proposed throughout the years. In this context, the best feature spaces are those that maximize *discrimination*, i.e. the separation between the different image classes to recognize. However, classical discriminant criteria such as *linear discriminant analysis* make very specific assumptions regarding class densities, e.g. Gaussianity, that are unrealistic for most problems involving real data. Recently, various authors have advocated the use of information theoretic measures for feature extraction or selection [15, 3, 9, 11, 1]. These can be seen as instantiations of the *the infomax principle*

of neural organization[1] proposed by Linsker [7], which also encompasses information theoretic approaches for independent component analysis and blind-source separation [2]. In the classification context, infomax recommends the selection of the feature transform that maximizes the mutual information (MI) between features and class labels.

While searching for the features that preserve the maximum amount of information about the class is, at an intuitive level, an appealing discriminant criteria, the infomax principle does not establish a direct connection to the ultimate measure of classification performance - the *probability of error* (PE). By noting that to maximize MI between features and class labels is the same as minimizing the entropy of labels given features, it is possible to establish a connection through Fano's inequality: that class-posterior entropy (CPE) is a lower bound on the PE [11, 4]. This connection is, however, weak in the sense that there is little insight on how tight the bound is, or if minimizing it has any relationship to minimizing PE. In fact, among all lower bounds on PE, it is not clear that CPE is the most relevant. An obvious alternative is the Bayes error (BE) which 1) is the tightest possible classifier-independent lower-bound, 2) is an intrinsic measure of the complexity of the discrimination problem and, 3) like CPE, depends on the feature transformation and class labels alone. Minimizing BE has been recently proposed for feature extraction in speech problems [10].

The main contribution of this paper is to show that the two strategies (infomax and minimum BE) are very closely related. In particular, it is shown that 1) CPE is a lower bound on BE and 2) this bound is tight, in the sense that the former is a good approximation to the latter. It follows that *infomax solutions are near-optimal in the minimum BE sense*. While for feature extraction both infomax and BE appear to be difficult to optimize directly, we show that infomax has clear computational advantages for feature selection, particularly in the context of the sequential procedures that are prevalent in the feature selection literature [6]. The analysis of some simple classification problems reveals that a quantity which plays an important role in infomax solutions is the marginal diversity: the average distance between each of the marginal class-conditional densities and their mean. This serves as inspiration to a generic principle for feature selection, *the principle of maximum marginal diversity* (MMD), that only requires marginal density estimates and can therefore be implemented with extreme computational simplicity. While heuristics that are close to the MMD principle have been proposed in the past, very little is known regarding their optimality.

In this paper we summarize the main results of a theoretical characterization of the problems for which the principle is guaranteed to be optimal in the infomax sense (see [13] for further details). This characterization is interesting in two ways. First, it shows that *there is a family of classification problems for which a near-optimal solution, in the BE sense, can be achieved with a computational procedure that does not involve combinatorial search*. This is a major improvement, from a computational standpoint, to previous solutions for which some guarantee of optimality (branch and bound search) or near optimality (forward or backward search) is available [6]. Second, when combined with recent studies on the statistics of biologically plausible image transformations [8, 5], it suggests that *in the context of visual recognition, MMD feature selection will lead to solutions that are optimal in the infomax sense*. Given the computational simplicity of the MMD principle, this is quite significant. We present experimental evidence in support of these two properties of MMD.

## 2 Infomax vs minimum Bayes error

In this section we show that, for classification problems, the infomax principle is closely related to the minimization of Bayes error. We start by defining these quantities.

**Theorem 1** *Given a classification problem with $M$ classes in a feature space $\mathcal{X}$, the decision function which minimizes the probability of classification error is the Bayes classifier $g^*(\mathbf{x}) = \arg\max_i P_{Y|\mathbf{X}}(i|\mathbf{x})$, where $Y$ is a random variable that assigns $\mathbf{x}$ to one of $M$ classes, and $i \in \{1, \ldots, M\}$. Furthermore, the PE is lower bounded by the* Bayes error

$$L^* = 1 - E_{\mathbf{x}}[\max_i P_{Y|\mathbf{X}}(i|\mathbf{x})], \tag{1}$$

*where $E_{\mathbf{x}}$ means expectation with respect to $P_{\mathbf{X}}(\mathbf{x})$.*

*Proof:* All proofs are omitted due to space considerations. They can be obtained by contacting the author.

**Principle 1 (infomax)** *Consider an $M$-class classification problem with observations drawn from random variable $\mathbf{Z} \in \mathcal{Z}$, and the set of feature transformations $T : \mathcal{Z} \to \mathcal{X}$. The best feature space is the one that maximizes the mutual information $I(Y; \mathbf{X})$ where $Y$ is the class indicator variable defined above, $\mathbf{X} = T(\mathbf{Z})$, and $I(Y; \mathbf{X}) = \sum_i \int p_{\mathbf{X},Y}(\mathbf{x}, i) \log \frac{p_{\mathbf{X},Y}(\mathbf{x},i)}{p_{\mathbf{X}}(\mathbf{x})p_Y(i)} d\mathbf{x}$ the mutual information between $\mathbf{X}$ and $Y$.*

It is straightforward to show that $I(\mathbf{X}, Y) = H(Y) - H(Y|\mathbf{X})$, where $H(\mathbf{X}) = -\int p_{\mathbf{X}}(\mathbf{x}) \log p_{\mathbf{X}}(\mathbf{x}) d\mathbf{x}$ is the entropy of $\mathbf{X}$. Since the class entropy $H(Y)$ does not depend on $T$, infomax is equivalent to the minimization of the CPE $H(Y|\mathbf{X})$. We next derive a bound that plays a central role on the relationship between this quantity and BE.

**Lemma 1** *Consider a probability mass function $\mathbf{p} = \{p_1, \ldots, p_M\}$ such that $0 \leq p_i \leq 1, \forall i$ and $\sum_i p_i = 1$. Then,*

$$(1 - \max_i p_i) \geq \frac{1}{\log M} H(\mathbf{p}) - \frac{\log(2M-1)}{\log M} + 1 \tag{2}$$

*where $H(\mathbf{p}) = -\sum_i p_i \log p_i$. Furthermore, the bound is tight in the sense that equality holds when*

$$p_j^* = \frac{M}{2M-1} \text{ and } p_k^* = \frac{1}{2M-1}, \forall k \neq j. \tag{3}$$

The following theorem follows from this bound.

**Theorem 2** *The BE of an $M$-class classification problem with feature space $\mathcal{X}$ and class indicator variable $Y$, is lower bounded by*

$$L_{\mathcal{X}}^*(M) \geq \frac{1}{\log M} H(Y|\mathbf{X}) - \frac{\log(2M-1)}{\log M} + 1, \tag{4}$$

*where $\mathbf{X} \in \mathcal{X}$ is the random vector from which features are drawn. When $M$ is large ($M \to \infty$) this bound reduces to $L_{\mathcal{X}}^*(M) \geq \frac{1}{\log M} H(Y|\mathbf{X})$.*

It is interesting to note the relationship between (4) and Fano's lower bound on the PE ($P_e \geq \frac{1}{\log M} H(Y|\mathbf{X}) - \frac{1}{\log M}$). The two bounds are equal up to an additive constant ($\frac{1}{\log M} \log \frac{2M}{2M-1}$) that quickly decreases to zero with the number of classes $M$. It follows that, at least when the number of classes is large, Fano's is really a lower bound on BE, not only on PE. Besides making this clear, Theorem 2 is a relevant contribution in two ways. First, since constants do not change the location of the bound's extrema, it shows that *infomax minimizes a lower bound on BE*. Second, unlike Fano's bound, it sheds considerable insight on the relationship between the extrema of the bound and those of the BE.

In fact, it is clear from the derivation of the theorem that, the only reason why the right-hand (RHS) and left-hand (LHS) sides of (4) differ is the application of (2). Figure 1

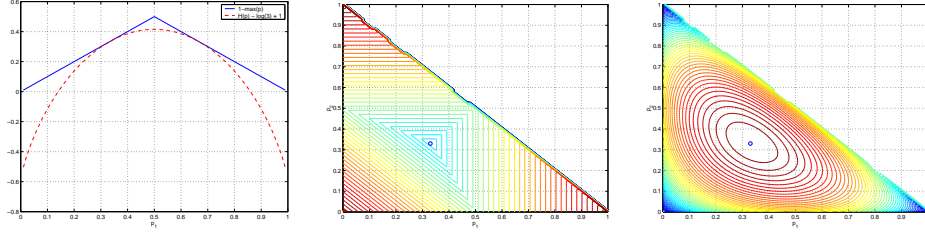

Figure 1: Visualization of (2). Left: LHS and RHS versus $p_1$ for $M = 2, (p_2 = 1 - p_1)$. Middle: contours of the LHS versus $(p_1, p_2)$ for $M = 3, (p_3 = 1 - p_1 - p_2)$. Right: same, for RHS.

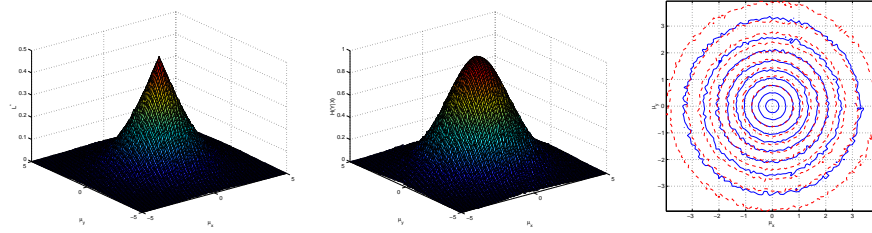

Figure 2: The LHS of (4) as an approximation to (1) for a two-class Gaussian problem where $P_{\mathbf{X}|Y}(\mathbf{x}|1) \sim N(\mathbf{0}, \mathbf{I})$ and $P_{\mathbf{X}|Y}(\mathbf{x}|2) \sim N(\mu, \mathbf{I})$. All plots are functions of $\mu$. Left: surface plot of (1). Middle: surface plot of the LHS of (4). Right: contour plots of the two functions.

shows plots of the RHS and LHS of this equation when $M \in \{2, 3\}$, illustrating three interesting properties. First, bound (2) is tight in the sense defined in the lemma. Second, the maximum of the LHS is co-located with that of the RHS. Finally, (like the RHS) the LHS is a concave function of $\mathbf{p}$ and increasing (decreasing) when the RHS is. Due to these properties, the LHS is a good *approximation* to the RHS and, consequently, the LHS of (4) a good approximation to its RHS. It follows that *infomax solutions will, in general, be very similar to those that minimize the BE* . This is illustrated by a simple example in Figure 2.

## 3   Feature selection

For feature extraction, both infomax and minimum BE are complicated problems that can only be solved up to approximations [9, 11, 10]. It is therefore not clear which of the two strategies will be more useful in practice. We now show that the opposite holds for feature selection, where the minimization of CPE is significantly simpler than that of BE. We start by recalling that, because the possible number of feature subsets in a feature selection problem is combinatorial, feature selection techniques rely on *sequential search* methods [6]. These methods proceed in a sequence of steps, each adding a set of features to the current best subset, with the goal of optimizing a given cost function[2]. We denote the current subset by $\mathbf{X}_c$, the added features by $\mathbf{X}_a$ and the new subset by $\mathbf{X}_n = (\mathbf{X}_a, \mathbf{X}_c)$.

**Theorem 3** *Consider an $M$-class classification problem with observations drawn from a random variable $\mathbf{Z} \in \mathcal{Z}$, and a feature transformation $T : \mathcal{Z} \rightarrow \mathcal{X}$. $\mathcal{X}$ is a infomax feature*

*space if and only if $\forall T' \neq T$*

$$\left\langle KL\left[P_{\mathbf{X}|Y}(\mathbf{x}|i)||P_{\mathbf{X}}(\mathbf{x})\right]\right\rangle_Y \geq \left\langle KL\left[P_{\mathbf{X}'|Y}(\mathbf{x}|i)||P_{\mathbf{X}'}(\mathbf{x})\right]\right\rangle_Y \tag{5}$$

*where $\mathbf{X} = T(\mathbf{Z})$, $\mathbf{X}' = T'(\mathbf{Z})$, $\langle f(i)\rangle_Y = \sum_i P_Y(i)f(i)$ denotes expectation with respect to the prior class probabilities and $KL[p||q] = \int p(\mathbf{x})\log\frac{p(\mathbf{x})}{q(\mathbf{x})}d\mathbf{x}$ is the Kullback-Leibler divergence between $p$ and $q$. Furthermore, if $\mathbf{X}_n = (\mathbf{X}_a, \mathbf{X}_c)$, the infomax cost function decouples into two terms according to*

$$\begin{aligned}
\left\langle KL\left[P_{\mathbf{X}_n|Y}(\mathbf{x}_n|i)||P_{\mathbf{X}_n}(\mathbf{x}_n)\right]\right\rangle_Y &= \left\langle KL\left[P_{\mathbf{X}_a|\mathbf{X}_c,Y}(\mathbf{x}_a|\mathbf{x}_c,i)||P_{\mathbf{X}_a|\mathbf{X}_c}(\mathbf{x}_a|\mathbf{x}_c)\right]\right\rangle_Y \\
&+ \left\langle KL\left[P_{\mathbf{X}_c|Y}(\mathbf{x}_c|i)||P_{\mathbf{X}_c}(\mathbf{x}_c)\right]\right\rangle_Y.
\end{aligned} \tag{6}$$

Equation (5) exposes the discriminant nature of the infomax criteria. Noting that $P_{\mathbf{X}}(\mathbf{x}) = < P_{\mathbf{X}|Y}(\mathbf{x}|i) >_Y$, it clearly favors feature spaces where each class-conditional density is as distant as possible (in the KL sense) from the average among all classes. This is a sensible way to quantify the intuition that optimal discriminant transforms are the ones that best separate the different classes. Equation (6), in turn, leads to an optimal rule for finding the features $\mathbf{X}_a$ to merge with the current optimal solution $\mathbf{X}_c$: the set which minimizes $\left\langle KL\left[P_{\mathbf{X}_a|\mathbf{X}_c,Y}(\mathbf{x}_a|\mathbf{x}_c,i)||P_{\mathbf{X}_a|\mathbf{X}_c}(\mathbf{x}_a|\mathbf{x}_c)\right]\right\rangle_Y$. The equation also leads to a straightforward procedure for updating the optimal cost once this set is determined. On the other hand, when the cost function is BE, the equivalent expression is

$$E_{\mathbf{X}_n}[\max_i P_{Y|\mathbf{X}_n}(i|\mathbf{x}_n)] = E_{\mathbf{X}_c}\left\{E_{\mathbf{X}_a|\mathbf{X}_c}[\max_i \frac{P_{\mathbf{X}_a|Y,\mathbf{X}_c}(\mathbf{x}_a|i,\mathbf{x}_c)}{P_{\mathbf{X}_a|\mathbf{X}_c}(\mathbf{x}_a|\mathbf{x}_c)}P_{Y|\mathbf{X}_c}(i|\mathbf{x}_c)]\right\}. \tag{7}$$

Note that the non-linearity introduced by the $\max$ operator, makes it impossible to express $E_{\mathbf{X}_n}[\max_i P_{Y|\mathbf{X}_n}(i|\mathbf{x}_n)]$ as a function of $E_{\mathbf{X}_c}[\max_i P_{Y|\mathbf{X}_c}(i|\mathbf{x}_c)]$. For this reason, *infomax is a better principle for feature selection problems than direct minimization of BE.*

## 4 Maximum marginal diversity

To gain some intuition for infomax solutions, we next consider the Gaussian problem of Figure 3. Assuming that the two classes have equal prior probabilities $(P_Y(1) = P_Y(2) = 1/2)$, the marginals $P_{X_1|Y}(x|1)$ and $P_{X_1|Y}(x|2)$ are equal and feature $X_1$ does not contain any useful information for classification. On the other hand, because the classes are clearly separated along the $x_2$ axis, feature $X_2$ contains all the information available for discriminating between them. The different discriminating powers of the two variables are reflected by the infomax costs: while $P_{X_1}(x) = P_{X_1|Y}(x|1) = P_{X_1|Y}(x|2)$ leads to $< KL[P_{X_1|Y}(x|i)||P_{X_1}(x)] >_Y = 0$, from $P_{X_2}(x) \neq P_{X_2|Y}(x|1) \neq P_{X_2|Y}(x|2)$ it follows that $< KL[P_{X_2|Y}(x|i)||P_{X_2}(x)] >_Y > 0$, and (5) recommends the selection of $X_2$. This is unlike energy-based criteria, such as principal component analysis, that would select $X_1$. The key advantage of infomax is that it emphasizes marginal diversity.

**Definition 1** *Consider a classification problem on a feature space $\mathcal{X}$, and a random vector $\mathbf{X} = (X_1, \ldots, X_n)$ from which feature vectors are drawn. Then, $\mathbf{md}(X_k) = < KL[P_{X_k|Y}(x|i)||P_{X_k}(x)] >_Y$ is the marginal diversity of feature $X_k$.*

The intuition conveyed by the example above can be easily transformed into a generic principle for feature selection.

**Principle 2 (Maximum marginal diversity)** *The best solution for a feature selection problem is to select the subset of features that leads to a set of maximally diverse marginal densities.*

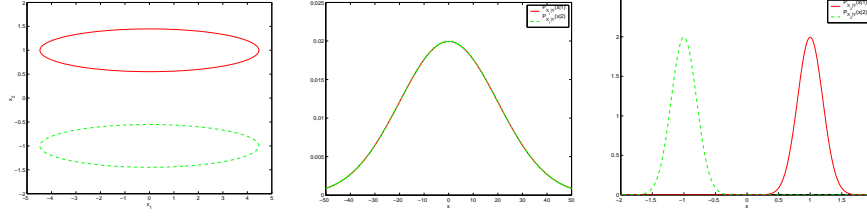

Figure 3: Gaussian problem with two classes $Y \in \{1, 2\}$, in the two-dimensions, $\mathbf{X} = (X_1, X_2)$. Left: contours of 65% probability. Middle: marginals for $X_1$. Right: marginals for $X_2$.

This principle has two attractive properties. First it is inherently discriminant, recommending the elimination of the dimensions along which the projections of the class densities are most similar. Second, it is straightforward to implement with the following algorithm.

**Algorithm 1 (MMD feature selection)** *For a classification problem with $n$ features $\mathbf{X} = (X_1, \ldots, X_n)$, $M$ classes $Y \in \{1, \ldots, M\}$ and class priors $P_Y(i) = p_i$ the following procedure returns the top $N$ MMD features.*

*- foreach feature $k \in \{1, \ldots, n\}$:*
  *\* foreach class $i \in \{1, \ldots, M\}$, compute an histogram estimate $\mathbf{h}_{k,i}$ of $P_{X_k|Y}(x|i)$,*
  *\* compute $\mathbf{h}_k = \frac{1}{M} \sum_i h_{k,i}$,*
  *\* compute the marginal diversity $\mathbf{md}(X_k) = \sum_i p_i \mathbf{h}_{k,i}^T \log(\mathbf{h}_{k,i}./\mathbf{h}_k)$, where both the log and division ./ are performed element-wise,*
*- order the features by decreasing diversity, i.e. find $\{k_1, \ldots, k_n\}$ such that $\mathbf{md}(X_{k_i}) \geq \mathbf{md}(X_{k_{i+1}})$, and return $\{X_{k_1}, \ldots, X_{k_N}\}$.*

In general, there are no guarantees that MMD will lead to the infomax solution. In [13] we seek a precise characterization of the problems where MMD is indeed equivalent to infomax. Due to space limitations we present here only the main result of this analysis, see [13] for a detailed derivation.

**Theorem 4** *Consider a classification problem with class labels drawn from a random variable $Y$ and features drawn from a random vector $\mathbf{X} = (X_1, \ldots, X_n)$ and let $\mathbf{X}^* = (X_1^*, \ldots, X_N^*)$ be the optimal feature subset of size $N$ in the infomax sense. If*

$$I(X_k^*; \mathbf{X}_{1,k-1}^*) = I(X_k^*; \mathbf{X}_{1,k-1}^*|Y), \forall k \in \{1, \ldots, N\} \tag{8}$$

*where $\mathbf{X}_{1,k-1}^* = \{X_1^*, \ldots, X_{k-1}^*\}$, the set $\mathbf{X}^*$ is also the optimal subset of size $N$ in the MMD sense. Furthermore,*

$$\left\langle KL \left[ P_{\mathbf{X}^*|Y}(\mathbf{x}|i) \| P_{\mathbf{X}^*}(\mathbf{x}) \right] \right\rangle_Y = \sum_{k=1}^{N} \mathbf{md}(X_k^*). \tag{9}$$

The theorem states that the MMD and infomax solutions will be identical when the mutual information between features is not affected by knowledge of the class label. This is an interesting condition in light of various recent studies that have reported the observationof consistent patterns of dependence between the features of various biologically plausible image transformations [8, 5]. Even though the details of feature dependence will vary from one image class to the next, these studies suggest that the coarse structure of the patterns of dependence between such features follow universal statistical laws that hold for all types of images. The potential implications of this conjecture are quite significant. First it implies

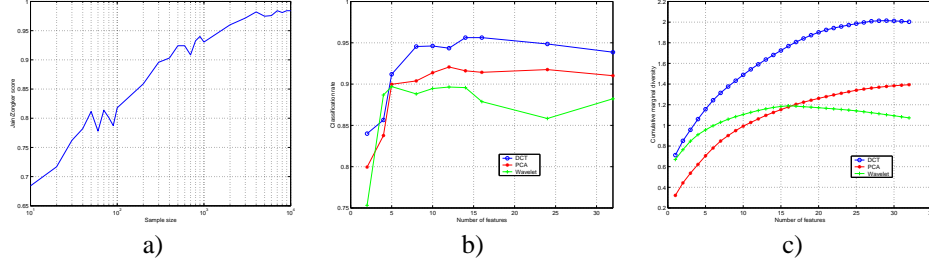

Figure 4: a) JZ score as a function of sample size for the two-class Gaussian problem discussed in the text, b) classification accuracy on Brodatz as a function of feature space dimension, and c) corresponding curves of cumulative marginal density (9). A linear trend was subtracted to all curves in c) to make the differences more visible.

that, in the context of visual processing, (8) will be approximately true and the MMD principle will consequently lead to solutions that are very close to optimal, in the minimum BE sense. Given the simplicity of MMD feature selection, this is quite remarkable. Second, it implies that when combined with such transformations, the marginal diversity is a close predictor for the CPE (and consequently the BE) achievable in a given feature space. This enables quantifying the goodness of the transformation without even having to build the classifier. See [13] for a more extensive discussion of these issues.

## 5 Experimental results

In this section we present results showing that 1) MMD feature selection outperforms combinatorial search when (8) holds, and 2) in the context of visual recognition problems, marginal diversity is a good predictor of PE. We start by reporting results on a synthetic problem, introduced by Trunk to illustrate the curse of dimensionality [12], and used by Jain and Zongker (JZ) to evaluate various feature selection procedures [6]. It consists of two Gaussian classes of identity covariance and means $\pm[1 \; \frac{1}{\sqrt{2}} \; \frac{1}{\sqrt{3}} \ldots \frac{1}{\sqrt{n}}]^T$ and is an interesting benchmark for feature selection because it has a clear optimal solution for the best subset of $d$ features (the first $d$) for any $d$.    JZ exploited this property to propose an automated procedure for testing the performance of feature selection algorithms across variations in dimensionality of the feature space and sample size. We repeated their experiments, simply replacing the cost function they used (Mahalanobis distance - MDist - between the means) by the marginal diversity.

Figure 4 a) presents the JZ score obtained with MMD as a function of the sample size. A comparison with Figure 5 of [6] shows that these results are superior to all those obtained by JZ, including the ones relying on branch and bound. This is remarkable, since branch and bound is guaranteed to find the optimal solution and the Mdist is inversely proportional to the PE for Gaussian classes. We believe that the superiority of MMD is due to the fact that it only requires estimates of the marginals, while the MDist requires estimates of joint densities and is therefore much more susceptible to the curse of dimensionality. Unfortunately, because in [6] all results are averaged over dimension, we have not been able to prove this conjecture yet. In any case, this problem is a good example of situations where, because (8) holds, MMD will find the optimal solution at a computational cost that is various orders of magnitude smaller than standard procedures based on combinatorial search (e.g. branch and bound).

Figures 4 b) and c) show that, for problems involving commonly used image transformations, marginal diversity is indeed a good predictor of classification accuracy. The figures

compare, for each space dimension, the recognition accuracy of a complete texture recognition system with the predictions provided by marginal diversity. Recognition accuracy was measured on the Brodatz texture database (112 texture classes) and a 64 dimensional feature space consisting of the coefficients of a multiresolution decomposition over regions of $8 \times 8$ pixels. Three transformations were considered: the discrete cosine transform, principal component analysis, and a three-level wavelet decomposition (see [14] for detailed description of the experimental set up). The classifier was based on Gauss mixtures and marginal diversity was computed with Algorithm 1. Note that the curves of cumulative marginal diversity are qualitatively very similar to those of recognition accuracy.

## Footnotes

[1]Under the infomax principle, the optimal organization for a complex multi-layered perceptual system is one where the information that reaches each layer is processed so that the maximum amount of information is preserved for subsequent layers.

[2]These methods are called *forward search* techniques. There is also an alternative set of *backward search* techniques, where features are successively removed from an initial set containing all features. We ignore the latter for simplicity, even though all that is said can be applied to them as well.

## References

[1] S. Basu, C. Micchelli, and P. Olsen. Maximum Entropy and Maximum Likelihood Criteria for Feature Selection from Multivariate Data. In *Proc. IEEE International Symposium on Circuits and Systems*, Geneva, Switzerland,2000.

[2] A. Bell and T. Sejnowski. An Information Maximisation Approach to Blind Separation and Blind Deconvolution. *Neural Computation*, 7(6):1129–1159, 1995.

[3] B. Bonnlander and A. Weigand. Selecting Input Variables using Mutual Information and Non-parametric Density Estimation. In *Proc. IEEE International ICSC Symposium on Artificial Neural Networks*, Tainan,Taiwan,1994.

[4] D. Erdogmus and J. Principe. Information Transfer Through Classifiers and its Relation to Probability of Error. In *Proc. of the International Joint Conference on Neural Networks*, Washington, 2001.

[5] J. Huang and D. Mumford. Statistics of Natural Images and Models. In *IEEE Computer Society Conference on Computer Vision and Pattern Recognition, Fort Collins, Colorado*, 1999.

[6] A. Jain and D. Zongker. Feature Selection: Evaluation, Application, and Small Sample Performance. *IEEE Trans. on Pattern Analysis and Machine Intelligence*, 19(2):153–158, February 1997.

[7] R. Linsker. Self-Organization in a Perceptual Network. *IEEE Computer*, 21(3):105–117, March 1988.

[8] J. Portilla and E. Simoncelli. Texture Modeling and Synthesis using Joint Statistics of Complex Wavelet Coefficients. In *IEEE Workshop on Statistical and Computational Theories of Vision, Fort Collins, Colorado*, 1999.

[9] J. Principe, D. Xu, and J. Fisher. Information-Theoretic Learning. In S. Haykin, editor, *Unsupervised Adaptive Filtering, Volume 1: Blind-Souurce Separation*. Wiley, 2000.

[10] G. Saon and M. Padmanabhan. Minimum Bayes Error Feature Selection for Continuous Speech Recognition. In *Proc. Neural Information Proc. Systems*, Denver, USA, 2000.

[11] K. Torkolla and W. Campbell. Mutual Information in Learning Feature Transforms. In *Proc. International Conference on Machine Learning*, Stanford, USA, 2000.

[12] G. Trunk. A Problem of Dimensionality: a Simple Example. *IEEE Trans. on Pattern. Analysis and Machine Intelligence*, 1(3):306–307, July 1979.

[13] N. Vasconcelos. Feature Selection by Maximum Marginal Diversity: Optimality and Implications for Visual Recognition. In *submitted*, 2002.

[14] N. Vasconcelos and G. Carneiro. What is the Role of Independence for Visual Regognition? In *Proc. European Conference on Computer Vision, Copenhagen, Denmark*, 2002.

[15] H. Yang and J. Moody. Data Visualization and Feature Selection: New Algorithms for Nongaussian Data. In *Proc. Neural Information Proc. Systems*, Denver, USA, 2000.
